# Multivariate Time Series Imputation with Generative Adversarial Networks

**Yonghong Luo**
College of Computer Science
Nankai University
Tianjin, China
luoyonghong@dbis.nankai.edu.cn

**Xiangrui Cai**
College of Computer Science
Nankai University
Tianjin, China
caixiangrui@dbis.nankai.edu.cn

**Ying Zhang** *
College of Computer Science
Nankai University
Tianjin, China
yingzhang@nankai.edu.cn

**Jun Xu**
School of Information
Renmin University of China
Beijing, China
junxu@ruc.edu.cn

**Xiaojie Yuan**
College of Computer Science
Nankai University
Tianjin, China
yuanxj@nankai.edu.cn

## Abstract

Multivariate time series usually contain a large number of missing values, which hinders the application of advanced analysis methods on multivariate time series data. Conventional approaches to addressing the challenge of missing values, including mean/zero imputation, case deletion, and matrix factorization-based imputation, are all incapable of modeling the temporal dependencies and the nature of complex distribution in multivariate time series. In this paper, we treat the problem of missing value imputation as data generation. Inspired by the success of Generative Adversarial Networks (GAN) in image generation, we propose to learn the overall distribution of a multivariate time series dataset with GAN, which is further used to generate the missing values for each sample. Different from the image data, the time series data are usually incomplete due to the nature of data recording process. A modified Gate Recurrent Unit is employed in GAN to model the temporal irregularity of the incomplete time series. Experiments on two multivariate time series datasets show that the proposed model outperformed the baselines in terms of accuracy of imputation. Experimental results also showed that a simple model on the imputed data can achieve state-of-the-art results on the prediction tasks, demonstrating the benefits of our model in downstream applications.

## 1 Introduction

The real world is filled with multivariate time series data such as network records, medical logs and meteorologic observations. Time series analysis is useful in many situations such as forecasting the stock price [22] and indicating fitness and diagnosis category of patients [7]. However, some of these time series are incomplete due to the broken of collective devices, the collecting errors and willful damages [15]. Besides, the time intervals of the observations in time series are not always fixed. Figure 1 and Figure 2 demonstrate the high missing rate of the Physionet [42] dataset. As time goes by, the maximum missing rate at each timestamp is always higher than 95%. We can also observe that most variables' missing rate are above 80% and the mean of the missing rate is 80.67%. The missing values in time series data make it hard to analyze and mine [14]. Therefore, the processing of missing values in time series has become a very important problem.

---

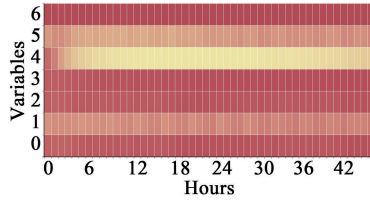

Figure 1: Missing rates in the Physionet dataset. The X-axis is the time. The Y-axis is the selected 7 variables. Redder the color, higher the missing rate.

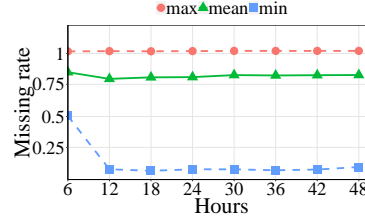

Figure 2: The lines stand for maximum, minimum and average missing rates at each hour. The global missing rate is 80.67%.

Usually there are two ways to handle the missing values of the dataset. Some researches try to directly model the dataset with missing values [48]. However, for every dataset, we need to model them separately. The second way is to impute the missing values to get the complete dataset and then use conventional methods to analyze the dataset. Existing missing values processing methods can be categorized into 3 classes. The very first one is case deletion methods [26, 43]. Its main idea is to discard the incomplete observations. However, these case deletion methods will ignore some important information. Additionally, the higher the missing rate, the worse the result [18]. The second kind of algorithms is simple imputation methods such as mean imputation, median imputation, and the most common value imputation. The main drawback of these statistical imputation methods is the lack of the utilization of the temporal information. The last kind of methods is machine learning based imputation algorithms [4, 19, 34]. These methods contain maximum likelihood Expectation-Maximization (EM) based imputation, KNN based imputation and Matrix Factorization based imputation. However, all of these methods rarely take into account the temporal relations between two observations.

In recent years, Goodfellow et al. [17] have introduced the generative adversarial networks (GAN) which learns the latent distribution of a dataset and is able to generate "real" samples from a random "noise". GAN has been successfully applied to face completion and sentence generation [5, 30, 33, 31, 13, 47]. However, before completion of the faces or generation of the sentences, these methods require the complete training dataset which is impossible in our scenario. There also exists a few works that use GAN to impute the missing values [46]. However, what these works focused is non-sequential dataset and they have not adopted pertinent measures to process the temporal relations. Hence, these algorithms can not be applied to the imputation of time series data well.

Inspired by the success of GAN in image imputation, we take the advantage of the adversarial model to generate and impute the original incomplete time series data. In order to learn the latent relationships between observations with non-fixed time lags, we propose a novel RNN cell called GRUI which can take into account the non-fixed time lags and fade the influence of the past observations determined by the time lags. In the first phase, by adopting the GRUI in the discriminator and generator in GAN, the well trained adversarial model can learn the distribution of the whole dataset, the implicit relationships between observations and the temporal information of the dataset. In the second phase, we train the input "noise" of the generator of the GAN so that the generated time series is as close as possible to the original incomplete time series and the generated data's probability of being real is the biggest. To the best of our knowledge, this is the first work that uses adversarial networks to impute time series dataset. We evaluate our method on a real-world medical dataset and a real-world meteorologic dataset. The results show the superiority of our approach compared to the baselines in terms of imputation accuracy. Our model is also superior to the baselines in prediction and regression tasks using the imputed datasets.

## 2 Method

Given a collection of multivariate time series with $d$ dimensions, one time series $\boldsymbol{X}$ observed in $\boldsymbol{T}=(t_0,\ldots,t_{n-1})$, is denoted by $\boldsymbol{X}=(\boldsymbol{x}_{t_0},\ldots,\boldsymbol{x}_{t_i},\ldots,\boldsymbol{x}_{t_{n-1}})^{\top} \in \mathbb{R}^{n \times d}$, where $\boldsymbol{x}_{t_i}$ is the $t_i$th observation of $\boldsymbol{X}$, and $x_{t_i}^j$ is the $j$th variable of $\boldsymbol{x}_{t_i}$. In the following example, $d$=4, $n$=3 and "none" is missing value,

$$\boldsymbol{X} = \begin{bmatrix} 1 & 6 & none & 9 \\ 7 & none & 7 & none \\ 9 & none & none & 79 \end{bmatrix}, T = \begin{bmatrix} 0 \\ 5 \\ 13 \end{bmatrix}.$$

The time series $\boldsymbol{X}$ is incomplete, we introduce the mask matrix $\boldsymbol{M} \in \mathbb{R}^{n \times d}$ to present whether the values of $\boldsymbol{X}$ exist or not, i.e., $M_{t_i}^j$=1, if $x_{t_i}^j$ exists, otherwise $M_{t_i}^j$=0.

In order to replace missing values in time series data with reasonable values, we first train a GAN based model to learn the distribution of the original time series dataset. In this custom GAN model, the generator which generates fake time series from a random input vector and the discriminator which distinguishes between fake data and real data, will achieve an equilibrium that not only increases the representative ability of the generator but also upgrades the discernment ability of the discriminator (see Figure 3). Next, we fix the network structure and optimize the input random vector of the generator so that the generated fake time series can best replace the missing values. In subsection 2.1, we show the details of the GAN Architecture. Subsection 2.2 demonstrates the method to impute the missing values.

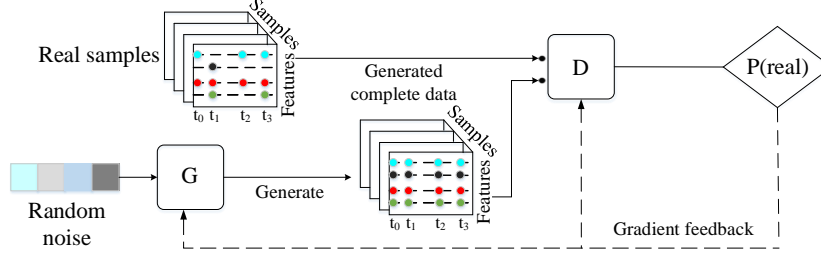

Figure 3: The structure of the proposed model.

## 2.1 GAN Architecture

A GAN is made up of a generator (G) and a discriminator (D). The G learns a mapping $G(\boldsymbol{z})$ that tries to map the random noise vector $\boldsymbol{z}$ to a realistic time series. The D tries to find a mapping $D(.)$ that tell us the input data's probability of being real. It is noteworthy that the input of the D contains the real but incomplete samples and the fake but complete samples generated by G. Because of the mode collapse problem [3], the traditional GAN is hard to train [20, 32, 37]. WGAN [3] is another training way of GAN which uses the Wasserstein distance that is easier to train than the original. WGAN can improve the stability of the learning stage, get out of the problem of mode collapse and make it easy for the optimization of the GAN model. In our method, we prefer WGAN [3] to traditional GAN. The following is the loss function of WGAN.

$$L_G = \mathbb{E}_{\boldsymbol{z} \sim P_g} \left[ -D(G(\boldsymbol{z})) \right] , \tag{1}$$

$$L_D = \mathbb{E}_{\boldsymbol{z} \sim P_g} \left[ D(G(\boldsymbol{z})) \right] - \mathbb{E}_{\boldsymbol{x} \sim P_r} \left[ D(\boldsymbol{x}) \right] . \tag{2}$$

When we design the detail structure of the GAN, we adopt Gated Recurrent Unit (GRU) [10], a state-of-the-art RNN cell, as the basic network of G and D. It is worth noting that, others RNN variants can also be used in this work, such as the Long Short-Term Memory (LSTM) [21] cell. However, the time lags between two consecutive valid observations vary a lot due to data incompleteness, which makes traditional GRU cell or LSTM cell not applicable to our senario. In order to effectively handle the irregular time lags and to learn the implicit information from the time intervals, we propose the GRUI cell based on GRU.

**GRUI**. To appropriate learn the distribution and characteristic of the original incomplete time series dataset, we find that, the time lag between two consecutive valid observations is always in changing because of the "none" values. The time lags between observations are very important since they follow an unknown nonuniform distribution. These changeable time lages remind us that the influence of the past observations should decay with time if the variable has been missing for a while. In order to fit this decayed influence of the past observations, we propose the Gated Recurrent Unit for data Imputation (**GRUI**) cell to model the temporal irregularity of the incomplete time series.

In order to record the time lag of two adjacent existent values of $\boldsymbol{X}$, we introduce the time lag matrix $\boldsymbol{\delta} \in \mathbb{R}^{n \times d}$ to record the time lag between current value and last valid value. The followings is the calculation way and calculated results of $\boldsymbol{\delta}$ of the sample dataset.

$$\delta_{t_i}^j = \begin{cases} t_i - t_{i-1}, & M_{t_{i-1}}^j == 1 \\ \delta_{t_{i-1}}^j + t_i - t_{i-1}, & M_{t_{i-1}}^j == 0 \ \& \ i > 0 \ ; \\ 0, & i == 0 \end{cases} \quad \boldsymbol{\delta} = \begin{bmatrix} 0 & 0 & 0 & 0 \\ 5 & 5 & 5 & 5 \\ 8 & 13 & 8 & 13 \end{bmatrix} .$$

We introduce a time decay vector $\boldsymbol{\beta}$ to control the influence of the past observations. Each value of the $\boldsymbol{\beta}$ should be bigger than zero and smaller than one, and the larger the $\boldsymbol{\delta}$, the smaller the decay vector. So we model the time decay vector $\boldsymbol{\beta}$ as a combination of $\boldsymbol{\delta}$:

$$\boldsymbol{\beta}_{t_i} = 1/e^{\max(\mathbf{0}, \boldsymbol{W}_\beta \boldsymbol{\delta}_{t_i} + \boldsymbol{b}_\beta)}, \tag{3}$$

where $\boldsymbol{W}_\beta$ and $\boldsymbol{b}_\beta$ are parameters that need to learn. We use the negative exponential formulation to make sure that $\boldsymbol{\beta}_{t_i} \in (\mathbf{0}, \mathbf{1}]$. Besides, in order to capture the interactions of the $\boldsymbol{\delta}$'s variables, we prefer full weight matrix to diagonal matrix for $\boldsymbol{W}_\beta$. After we have got the decay vector, we update the GRU hidden state $\boldsymbol{h}_{t_{i-1}}$ by element-wise multiplying the decay factor $\boldsymbol{\beta}$. Since we have used the batch normalization [24] technology, the hidden state $\boldsymbol{h}$ is smaller than 1 with a high probability. We choose multiplicative decay way rather than some other decay ways such as $\boldsymbol{h}^\beta$. The update functions of GRUI are:

$$\boldsymbol{h}'_{t_{i-1}} = \boldsymbol{\beta}_{t_i} \odot \boldsymbol{h}_{t_{i-1}}, \tag{4}$$

$$\boldsymbol{\mu}_{t_i} = \sigma(\boldsymbol{W}_\mu \left[\boldsymbol{h}'_{t_{i-1}}, \boldsymbol{x}_{t_i}\right] + \boldsymbol{b}_\mu), \qquad\qquad \boldsymbol{r}_{t_i} = \sigma(\boldsymbol{W}_r \left[\boldsymbol{h}'_{t_{i-1}}, \boldsymbol{x}_{t_i}\right] + \boldsymbol{b}_r), \tag{5}$$

$$\tilde{\boldsymbol{h}}_{t_i} = tanh(\boldsymbol{W}_{\tilde{h}} \left[\boldsymbol{r}_{t_i} \odot \boldsymbol{h}'_{t_{i-1}}, \boldsymbol{x}_{t_i}\right] + \boldsymbol{b}_{\tilde{h}}), \qquad \boldsymbol{h}_{t_i} = (\mathbf{1} - \boldsymbol{\mu}_{t_i}) \odot \boldsymbol{h}_{t'_{i-1}} + \boldsymbol{\mu}_{t_i} \odot \tilde{\boldsymbol{h}}_{t_i}, \tag{6}$$

where $\boldsymbol{\mu}$ is update gate, $\boldsymbol{r}$ is reset gate, $\tilde{\boldsymbol{h}}$ is candidate hidden state, $\sigma$ is the sigmoid activation function, $\boldsymbol{W}_{\tilde{h}}$, $\boldsymbol{W}_r$, $\boldsymbol{W}_\mu$, $\boldsymbol{b}_\mu$, $\boldsymbol{b}_r$ and $\boldsymbol{b}_{\tilde{h}}$ are training parameters and $\odot$ is an element-wise multiplication.

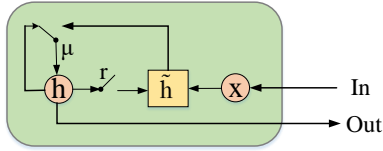

Figure 4: GRU cell.

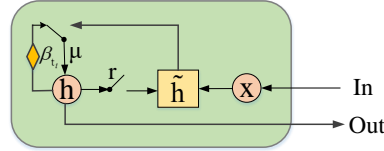

Figure 5: GRUI cell.

**D and G structure**. The D is first composed by a GRUI layer to learn the incomplete or complete time series. Then a full-connection layer is stacked on the top of the last hidden state of GRUI. To prevent overfit, we adopt dropout [44] to the full-connection layer. When we feed original incomplete real time series into D, values at one row of $\boldsymbol{\delta}$ are not the same. When we feed fake time series generated by G, values in each row of $\boldsymbol{\delta}$ are the same (because there is no missing value). We want to make sure that the time lags of the generated samples are the same as those of the original samples, so the G is also made up of a GRUI layer and a full-connection layer. The G is a self-feed network, it means that the current output of the G will be fed into the next iteration of the same cell. The very first input of G is the random noise vector $\boldsymbol{z}$ and every row of the $\boldsymbol{\delta}$ of fake sample is a constant value. That batch normalization [24] is applied both to G and D.

## 2.2 Missing Values Imputation by GAN

From the GAN architecture, we can know that, the generator G can learn a mapping $G(\boldsymbol{z}) = \boldsymbol{z} \mapsto \boldsymbol{x}$ that maps the random noise vector $\boldsymbol{z}$ to a complete time series which contains no missing value. However, the problem is the random noise vector $\boldsymbol{z}$ is randomly sampled from a latent space, e.g., Gaussian distribution. It means that, the generated samples may change a lot with the changing of the input random noise $\boldsymbol{z}$. Although the generated samples obey the distribution of the original samples, the distance between the generated samples and the original samples may also be large. In other words, the degree of similarity between $\boldsymbol{x}$ and $G(\boldsymbol{z})$ is not large enough. For example, the original incomplete time series contains two classes, and the G learns a distribution that can fit these two classes very well. Given a incomplete sample $\boldsymbol{x}$ and a random input vector $\boldsymbol{z}$, the $G(\boldsymbol{z})$ may belong to the opposite class of $\boldsymbol{x}$, this is not what we want. Although the $G(\boldsymbol{z})$ may belong to the true class, the similarity of samples within a class could also be large.

For any incomplete time series $\boldsymbol{x}$, we try to find a best vector $\boldsymbol{z}$ from the latent input space so that the generated sample $G(\boldsymbol{z})$ is most similar to $\boldsymbol{x}$. How to replace the missing values with the most reasonable values? Inspired by [41], we introduce a way to measure the degree of imputation fitness. We define a two-part loss function to evaluate the fitness of imputation. The first part of this loss function is the *masked reconstruction loss*. It means that the generated sample $G(\boldsymbol{z})$ should be

close enough to the original incomplete time series $x$. The another part of this loss function is the *discriminative loss*. This part forces the generated sample $G(z)$ as real as possible. The following paragraphs will describe the *masked reconstruction loss* and the *discriminative loss* in details.

**Masked Reconstruction Loss**. The *masked reconstruction loss* is defined by the masked squared errors between the original sample $x$ and the generated sample $G(z)$. It is noteworthy that we only calculate the non-missing part of the data.

$$L_r(z) = ||X \odot M - G(z) \odot M||_2 . \tag{7}$$

**Discriminative Loss**. The *discriminative loss* stands for the generated sample $G(z)$'s degree of authenticity. It is based on the output of the discriminator D which represents the confidence level of the input sample $G(z)$'s being real. We feed the noise vector $z$ into G, then we get the generated sample $G(z)$, next, we feed $G(z)$ into D, finally we get the *discriminative loss*.

$$L_d(z) = -D(G(z)) . \tag{8}$$

**Imputation Loss**. We define the *imputation loss* to optimize the random noise vector $z$. The *imputation loss* is a combination of the *masked reconstruction loss* and the *discriminative loss*.

$$L_{imputation}(z) = L_r(z) + \lambda L_d(z) , \tag{9}$$

where $\lambda$ is a hyper-parameter that controls the proportion between the *masked reconstruction loss* and the *discriminative loss*.

For each original time series $x$, we randomly sample a $z$ from the Gaussian distribution with zero mean and unit variance and feed it into the well trained generator G to get $G(z)$. Then we begin to train the noise $z$ with the loss function $L_{imputation}(z)$ by back propagation method. After the *imputation loss* converging to the optimal solution, we replace the missing values of $x$ with the generated $G(z)$ just as the following equation shows,

$$x_{imputed} = x \odot M + (1 - M) \odot G(z) . \tag{10}$$

## 3   Experiments

We evaluate the proposed method in two real-world datasets which include a medical dataset and a air quality dataset. In order to demonstrate the imputation results of the proposed method, we compare our algorithm with simple imputation methods, matrix factorization based imputation method and KNN based imputation method. We also compare our GAN based imputation method against some other baselines in the prediction and regression tasks.

### 3.1   Datasets and Tasks

**Physionet Challenge 2012 dataset (PhysioNet)**. The Physionet dataset is a public electronic medical record dataset that comes from the *PhysioNet Challenge 2012* [42]. This dataset consists of records from 4,000 intensive care unit (ICU) stays. Every ICU stay is a roughly 48 hours time series with 41 variables such as age, weight, albumin, heart-rate, glucose, etc. One task of the *PhysioNet Challenge 2012* is the mortality prediction task that predicts whether the patient dies in the hospital. There are 554 (13.85%) patients with positive mortality label. This task is a binary classification problem with non-balance dataset, so the AUC score is used to judge the performance of the classifier. Because of the lack of complete dataset, the direct evaluation of missing values filling accuracy is impossible. Therefore, we use the mortality prediction results calculated by the same classifier but trained on different imputed datasets to determine the performance of imputation methods. Machine learning methods must have enough training dataset to learn the potential relation between samples. We do not use the dataset processed by case deletion methods to train the classifier when we use the PhysioNet dataset because of its high missing rate (80.67%).

**KDD CUP 2018 Dataset (KDD)**. The KDD CUP 2018 dataset is a public air quality dataset that comes from the *KDD CUP Challenge 2018* [11]. KDD dataset contains the historical air quality data of Beijing. We select 11 common air quality and weather data observatories for our experiments. Each observatory owns records observed every one hour from January 1, 2017 to December 30, 2017. The records have total 12 variables which include PM2.5 (ug/m3), PM10 (ug/m3), CO (mg/m3), weather, temperature and so on. We split this dataset for every 48 hours just like the PhysioNet dataset, then

we get about 182 time series. For the split dataset, we conduct two tasks as the following described. 1) *Time series imputation task*: For every 48 hours length time series, we randomly discard $p$ percent of the dataset. Then we fill the missing values and calculate the imputation accuracy, where $p \in \{20, 30, \ldots, 90\}$. The imputation accuracy is defined as the mean squared error (MSE) between original values and imputed values. 2) *Air quality prediction task*: For every 48 hours length time series, we randomly discard 50 percent of the dataset. Then we predict the mean air quality of the next 6 hours. Just like what we did previously, we use the air quality prediction results calculated by the same regression model but trained on different imputed datasets to determine the performance of imputation methods.

| Dataset | # of Features | # of Samples | Missing Rate |
|---------|---------------|--------------|--------------|
| Physionet | 41 | 4000 | 80.67% |
| KDD | 132 | 182 | 1% |

Table 1: Dataset Statistics.

## 3.2 Training Settings

**Network details and training strategies**. The discriminator consists of a GRUI layer and a full-connection layer. We feed the real incomplete time series $x$, the fake but complete time series $G(z)$ and their corresponding $\delta$ into GRUI layer. Then the the last hidden state of GRUI layer will be fed into full-connection layer with a dropout to get the discriminator's output. The generator is a self-feed network that consists of a GRUI layer and a full-connection layer too. The current hidden state of GRUI layer is fed into full-connection layer with a dropout, then the output of full-connection layer will be treated as the input of the next iteration. All the outputs of full-connection layer are concatenated and batch normalized into the $G(z)$. The very first input of the generator is the random noise $z$. Before the training of the GAN, the generator is pretrained for some epochs with a squared error loss for predicting the next value in the training time series. For the PhysioNet dataset, the input dimension is 41 (we use all the variables of the PhysioNet dataset), batch size is 128, the hidden units number in GRUI of G and D is 64 and the dimension of random noise is also 64. For the KDD dataset, the input dimension is 132 (11 observatories $\times$ 12 variables), the batch size is 16, the number of hidden units in GRUI of G and D is 64 and the dimension of $z$ is 256.

**Comparative Methods**. When it is feasible to directly evaluate the imputation accuracy (task 1 of the KDD dataset), we compare the proposed method with simple imputation methods, the matrix factorization imputation method and the KNN imputation method. If it is impracticable to get the complete dataset, we use two tasks to indirectly measure the imputation accuracy. *1) Classification task (mortality prediction task)*: we use different datasets imputed by proposed method and some other methods to train logistic regression classifier, SVM classifier, random forest classifier and RNN classifier. Then we indirectly compare the filling accuracy of these methods. *2) Regression task (air quality prediction task)*: we use datasets imputed by different imputation methods to train linear regression model, decision tree regression model, random forest regression model and RNN based regression model. Then we indirectly compare the filling accuracy of these methods.

## 3.3 Results

**Experimental results on Physionet dataset**. For the PhysioNet dataset, we can not access the complete samples. Therefore, we measure the filling accuracies of our proposed method and some other imputation methods indirectly. The hyper-parameters of our method are: the train epochs is 30, pretrain epochs is 5, learning rate is 0.001, $\lambda$ is 0.15 and the number of optimization iterations of the *imputation loss* is 400. Figure 6 is the comparison results of the classification task (mortality prediction task). We first complete the dataset by filling last value, zero value, mean value and GAN generated value. The standardization of input dataset is conducted when we impute the missing values with mean value, last value and GAN generated value. If we also conduct standardization on zero value imputation, the zero value imputation will be same as the mean imputation. So we do not standardize the input dataset when we impute with zero value. We train the logistic regression classifier, SVM (with RBF kernel, Linear kernel, Poly kernel and Sigmoid kernel) classifiers, random forest classifier and RNN classifier on these above imputed complete datasets to indirectly compare the filling accuracy of these filling methods. The RNN classifier is composed by a GRUI layer that

processes complete time series and a full-connection layer that outputs classification results. We can see that, except for the SVM classifier with RBF kernel, the classifiers trained on dataset imputed by proposed method always gain the best AUC score. These results can prove the success of GAN based imputation method indirectly because of the lack of complete dataset. It is worth noting that, we achieve the new state-of-the-art mortality prediction result with AUC score of 0.8603 by using the dataset imputed by the GAN based imputation method, while the previous state-of-the-art AUC score is 0.848 [25]. Table 2 is the detail description of mortality prediction task results produced by different methods.

| Model | Result |
|---|---|
| Neural Network model called GRUD [7] | 0.8424 |
| Hazard Markov Chain model [29] | 0.8381 |
| Regularized Logistic Regression model [25] | 0.848 |
| GAN based imputation & RNN model | **0.8603** |

Table 2: The AUC score of the mortality prediction task on the Physionet dataset. The RNN model that uses the dataset imputed by our method achieves the highest AUC score.

**Experimental results on KDD dataset**. Table 3 shows the comparison results between the proposed GAN based method and some other imputation methods which include imputation method that uses the last observed value to impute missing values (last imputation), method that uses mean value to fill missing values (mean imputation), KNN based method and matrix factorization based method. Before the starting of the experiments, we have standardized the input dataset. Therefore, filling zero value is the same as filling mean value. The hyper-parameters of our method are: the train epochs is 25, pretrain epochs is 20, learning rate is 0.002, $\lambda$ is 0.0 and the number of optimization iterations of the *imputation loss* is 800.

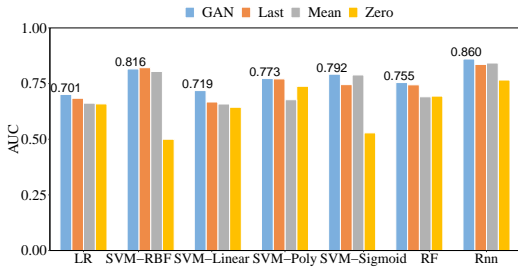

Figure 6: The AUC score of mortality prediction by different classification models trained on different imputed datasets.

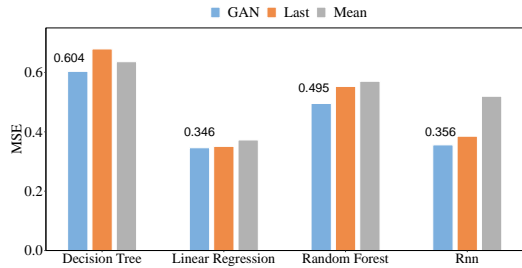

Figure 7: The MSE of air quality prediction by different regression models trained on different imputed datasets.

The first column of the table 3 is the missing-rate which indicates there are how many percent missing values in the dataset. The remaining columns are the mean squared errors (MSE) of the corresponding imputation methods. We can see that, with most missing-rates of the dataset, the proposed method owns the best filling accuracy. This is because the proposed GAN based method can automatically learn the temporal relationship of the same sample, the similarities between the similar samples, the association rules of two variables and the distribution of the dataset. By this way, the proposed GAN based method can fill the missing holes with the most reasonable values.

Figure 7 is the experimental results of the regression task. We use the KDD dataset with 50% percent missing values. Just like the settings of the classification task, we first fill the missing values. Then we train some regression models that include decision tree model, linear regression model, random forest model and RNN regression model. The RNN regression model is also made up of a GRUI layer and a full-connection layer. The hyper-parameters are the same as direct comparison. Because we have standardized the input dataset, zero filling is the same as mean filling. Figure 7 shows that the regression model trained with dataset which is imputed by the proposed method always gains the minimum MSE value. These results prove the success of GAN based imputation method.

| Missing-rate | Last filling | Mean filling | KNN filling | MF filling | GAN filling |
|---|---|---|---|---|---|
| 90% | 2.870 | **1.002** | 1.243 | 1.196 | 1.018 |
| 80% | 1.689 | 0.937 | 0.873 | 0.860 | **0.837** |
| 70% | 1.236 | 0.935 | 0.852 | 0.805 | **0.780** |
| 60% | 1.040 | 0.973 | 0.856 | 0.834 | **0.803** |
| 50% | 0.990 | 0.923 | 0.798 | 0.772 | **0.743** |
| 40% | 0.901 | 0.914 | 0.776 | 0.787 | **0.753** |
| 30% | 0.894 | 0.907 | 0.803 | 0.785 | **0.780** |
| 20% | 1.073 | 0.916 | 0.892 | 0.850 | **0.844** |

Table 3: The MSE results of the proposed method and other imputation methods on the KDD dataset. In most cases, the proposed method owns the best imputation accuracy.

**Comparison with GAN using a non-modified GRU.** We have also compared the proposed method with a GAN that use a non-modified GRU. In this situation, we do not take the advantage of the time interval information and then treat the time series as fixed interval data. So we do not model the time decay vector $\beta$ to control the influence of the past observations. We find that, with a non-modified GRU, the final AUC score of the Physionet dataset is 0.8029 while the GRUI is 0.8603. At the mean time, Table 4 shows the advantages of the GRUI cell tested on the KDD dataset. We can see that with the damping of the hidden state, the final performance of the imputation will increase a lot in all situations. The reason is that our model can learn and make use of the flexible time lags of the dataset and then produces better results than a non-modified GRU cell.

| Missing-rate | 90% | 80% | 70% | 60% | 50% | 40% | 30% | 20% |
|---|---|---|---|---|---|---|---|---|
| GRU | 1.049 | 0.893 | 0.841 | 0.823 | 0.794 | 0.767 | 0.820 | 0.849 |
| GRUI | **1.018** | **0.837** | **0.780** | **0.803** | **0.743** | **0.753** | **0.780** | **0.844** |

Table 4: The MSE comparison of a GAN with GRU and a GAN with GRUI on KDD dataset.

## 3.4 Discussions

**The proportion between discriminative loss and masked reconstruction loss.** In this part, we investigate the most influential hyper-parameter $\lambda$. Figure 8 and 9 show the impact of the $\lambda$, that is, the impact of the proportion between *discriminative loss* and *masked reconstruction loss*. We sample 13 values from 0.0 to 16.0 for $\lambda$ and compare the experimental results of these varied $\lambda$. When we perform the regression task on KDD dataset, we can conclude that with the growth of $\lambda$, the MSE of the KDD dataset grows near-linearly. It can be interpreted that the *masked reconstruction loss* dominates the *imputation loss* and the *discriminative loss* helps a little for the regression task on KDD dataset. The classification task results on PhysioNet dataset show that, the AUC score is small when the $\lambda$ is 0.0, and the AUC score reaches the maximum at the point of 0.15, then it decreases over the growth of $\lambda$. This phenomenon shows that the *discriminative loss* helps a lot for the classification task on PhysioNet dataset, especially with the $\lambda$ value of 0.15.

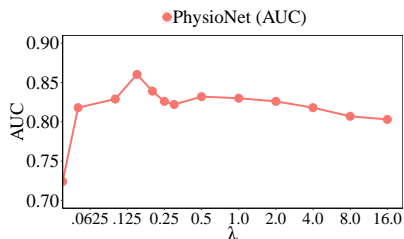

Figure 8: The influence of $\lambda$ in classification task. AUC score reaches the maximum at $\lambda =$ 0.15.

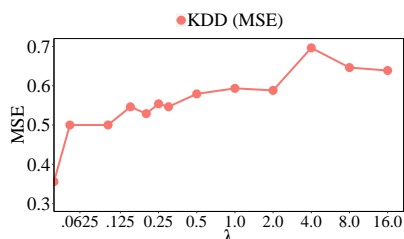

Figure 9: The influence of $\lambda$ in regression task. MSE reaches the minimum at $\lambda = 0.0$.

## 4 Related Work

This part will introduce the related works about missing value processing methods and generative adversarial networks.

### 4.1 Missing Value Processing Methods

The presence of missing values in datasets will significantly degrade the data analyses results [8]. In order to deal with the missing values in datasets, researchers have proposed many missing data handling methods in recent years. These methods can be classified into case **deletion based** methods, **simple imputation** methods and **machine learning based** imputation methods.

**Deletion based** methods erase all observations/records with missed values, including Listwise deletion [45] and Pairwise deletion [35]. The common drawback of the deletion methods is the loss of power when the missing rate is large enough (i.e. bigger than 5%) [18].

**Simple imputation** algorithms impute the missing values with some statistical attributes, such as replace missing value with mean value [27], impute with median value [1], impute with most common value [12] and complete the dataset with last observed valid value [2].

**Machine learning based** imputation methods include maximum likelihood Expectation-Maximization (EM) based imputation [38], K-Nearest Neighbor (KNN) based imputation [40], Matrix Factorization (MF) based imputation and Neural Network (NN) based imputation. The EM imputation algorithm is made up of the "expectation" step and the "maximization" step which iteratively updates model parameters and imputed values so that the model best fits the dataset. The KNN based imputation method uses the mean value of the $k$ nearest samples to impute missing values. The MF based imputation factorizes the incomplete matrix into low-rank matrices 'U' and 'V' solved by gradient descent algorithm, with a L1 sparsity penalty on the elements of 'U' and a L2 penalty on the elements of 'V'. Neural Network based imputation [16] uses the numerous parameters of the neural network to learn the distribution of train dataset and then fills the missing values.

### 4.2 Generative Adversarial Networks

At the year of 2014, Goodfellow et al [17] introduced the generative adversarial networks (GAN), which is a framework for estimating generative model via an adversarial process. The generative adversarial networks is made up of two components: a generator and a discriminator. The generator tries to fool the discriminator by generating fake samples from a random "noise" vector. The discriminator tries to distinguish between fake and real samples, i.e., to produce the probability that a sample comes from real datasets rather than the generator. However, the traditional GAN is hard to train, WGAN [3] is another training way of GAN, WGAN can improve the stability of learning and get out of the problem of mode collapse.

Many works have shown that the well trained GAN can produce realistic images in computer vision field [9, 23, 28, 36]. GAN is also successfully used to complete faces [5, 30, 33, 31]. Only A few works has introduced GAN into sequences generating field [6, 13, 39, 47], such as SeqGAN [47] and MaskGAN [13]. However, these works are not suitable for missing values imputation field. That is, before the generating of the sequences, these methods require the complete train dataset which is impossible in our scenario, yet our model needn't complete train datasets. Besides, most of GAN based sequence generation methods produce new samples from a random "noise" vector. With the changing of the random "noise" vector, the generated samples will change a lot. However, the data imputation task requires the imputed value as close as possible to the original incomplete data. There also exists a few work that uses GAN to impute the missing values such as GAIN [46]. The drawback of GAIN is the lack of consideration for the imputation of time series.

## 5 Conclusion

In this paper, we propose a novel generative adversarial networks for data imputation. In order to learn the unfixed time lags of two observed values, a modified GRU cell (called GRUI) is proposed for processing the incomplete time series. After the training of the GAN model with GRUI cell, the "noise" input vector of the generator is trained and generating reasonable values for imputation. In this way, the temporal relationships, the inner-class similarities, and the distribution of the dataset can be automatically learned under the adversarial architecture. Experimental results show that our method can outperform the baselines in terms of accuracy of missing value imputation, and has benefits for downstream applications.

## 6 Acknowledgements

We thank the reviewers for their constructive comments. We also thank Zhicheng Dou for his helpful suggestions. This research is supported by National Natural Science Foundation of China (No. 61772289 and No.61872338).

## References

[1] Edgar Acuna and Caroline Rodriguez. The treatment of missing values and its effect on classifier accuracy. In *Classification, clustering, and data mining applications*, pages 639–647. Springer, 2004.

[2] Mehran Amiri and Richard Jensen. Missing data imputation using fuzzy-rough methods. *Neurocomputing*, 205:152–164, 2016.

[3] Martin Arjovsky, Soumith Chintala, and Léon Bottou. Wasserstein generative adversarial networks. In *International Conference on Machine Learning*, pages 214–223, 2017.

[4] Gustavo EAPA Batista and Maria Carolina Monard. An analysis of four missing data treatment methods for supervised learning. *Applied artificial intelligence*, 17(5-6):519–533, 2003.

[5] Ashish Bora, Eric Price, and Alexandros G Dimakis. Ambientgan: Generative models from lossy measurements. In *International Conference on Learning Representations (ICLR)*, 2018.

[6] Tong Che, Yanran Li, Ruixiang Zhang, R Devon Hjelm, Wenjie Li, Yangqiu Song, and Yoshua Bengio. Maximum-likelihood augmented discrete generative adversarial networks. *arXiv preprint arXiv:1702.07983*, 2017.

[7] Zhengping Che, Sanjay Purushotham, Kyunghyun Cho, David Sontag, and Yan Liu. Recurrent neural networks for multivariate time series with missing values. *Scientific reports*, 8(1):6085, 2018.

[8] Jehanzeb R Cheema. A review of missing data handling methods in education research. *Review of Educational Research*, 84(4):487–508, 2014.

[9] Xi Chen, Yan Duan, Rein Houthooft, John Schulman, Ilya Sutskever, and Pieter Abbeel. Infogan: Interpretable representation learning by information maximizing generative adversarial nets. In *Advances in Neural Information Processing Systems*, pages 2172–2180, 2016.

[10] Kyunghyun Cho, Bart Van Merriënboer, Caglar Gulcehre, Dzmitry Bahdanau, Fethi Bougares, Holger Schwenk, and Yoshua Bengio. Learning phrase representations using rnn encoder-decoder for statistical machine translation. *arXiv preprint arXiv:1406.1078*, 2014.

[11] KDD Cup. Available on: http://www.kdd.org/kdd2018/, 2018.

[12] A Rogier T Donders, Geert JMG Van Der Heijden, Theo Stijnen, and Karel GM Moons. A gentle introduction to imputation of missing values. *Journal of clinical epidemiology*, 59(10):1087–1091, 2006.

[13] William Fedus, Ian Goodfellow, and Andrew M Dai. Maskgan: Better text generation via filling in the _. *arXiv preprint arXiv:1801.07736*, 2018.

[14] Pedro J García-Laencina, Pedro Henriques Abreu, Miguel Henriques Abreu, and Noémia Afonoso. Missing data imputation on the 5-year survival prediction of breast cancer patients with unknown discrete values. *Computers in biology and medicine*, 59:125–133, 2015.

[15] Pedro J García-Laencina, José-Luis Sancho-Gómez, and Aníbal R Figueiras-Vidal. Pattern classification with missing data: a review. *Neural Computing and Applications*, 19(2):263–282, 2010.

[16] Iffat A Gheyas and Leslie S Smith. A neural network-based framework for the reconstruction of incomplete data sets. *Neurocomputing*, 73(16-18):3039–3065, 2010.

[17] Ian Goodfellow, Jean Pouget-Abadie, Mehdi Mirza, Bing Xu, David Warde-Farley, Sherjil Ozair, Aaron Courville, and Yoshua Bengio. Generative adversarial nets. In *Advances in neural information processing systems*, pages 2672–2680, 2014.

[18] John W Graham. Missing data analysis: Making it work in the real world. *Annual review of psychology*, 60:549–576, 2009.

[19] Trevor Hastie, Rahul Mazumder, Jason D Lee, and Reza Zadeh. Matrix completion and low-rank svd via fast alternating least squares. *Journal of Machine Learning Research*, 16:3367–3402, 2015.

[20] Martin Heusel, Hubert Ramsauer, Thomas Unterthiner, Bernhard Nessler, Günter Klambauer, and Sepp Hochreiter. Gans trained by a two time-scale update rule converge to a nash equilibrium. *arXiv preprint arXiv:1706.08500*, 2017.

[21] Sepp Hochreiter and Jürgen Schmidhuber. Long short-term memory. *Neural computation*, 9(8):1735–1780, 1997.

[22] Tsung Jung Hsieh, Hsiao Fen Hsiao, and Wei Chang Yeh. Forecasting stock markets using wavelet transforms and recurrent neural networks: An integrated system based on artificial bee colony algorithm. *Applied Soft Computing Journal*, 11(2):2510–2525, 2011.

[23] Xun Huang, Yixuan Li, Omid Poursaeed, John Hopcroft, and Serge Belongie. Stacked generative adversarial networks. In *IEEE Conference on Computer Vision and Pattern Recognition (CVPR)*, volume 2, page 4, 2017.

[24] Sergey Ioffe and Christian Szegedy. Batch normalization: Accelerating deep network training by reducing internal covariate shift. *arXiv preprint arXiv:1502.03167*, 2015.

[25] Alistair EW Johnson, Andrew A Kramer, and Gari D Clifford. Data preprocessing and mortality prediction: The physionet/cinc 2012 challenge revisited. In *Computing in Cardiology Conference (CinC), 2014*, pages 157–160. IEEE, 2014.

[26] Jiri Kaiser. Dealing with missing values in data. *Journal of systems integration*, 5(1):42, 2014.

[27] Mehmed Kantardzic. *Data mining: concepts, models, methods, and algorithms*. John Wiley & Sons, 2011.

[28] Christian Ledig, Lucas Theis, Ferenc Huszár, Jose Caballero, Andrew Cunningham, Alejandro Acosta, Andrew Aitken, Alykhan Tejani, Johannes Totz, Zehan Wang, et al. Photo-realistic single image super-resolution using a generative adversarial network. *arXiv preprint*, 2016.

[29] Dae Hyun Lee and Eric Horvitz. Predicting mortality of intensive care patients via learning about hazard. In *AAAI*, pages 4953–4954, 2017.

[30] Yijun Li, Sifei Liu, Jimei Yang, and Ming-Hsuan Yang. Generative face completion. In *Proceedings of the IEEE Conference on Computer Vision and Pattern Recognition*, volume 1, page 6, 2017.

[31] Pengpeng Liu, Xiaojuan Qi, Pinjia He, Yikang Li, Michael R Lyu, and Irwin King. Semantically consistent image completion with fine-grained details. *arXiv preprint arXiv:1711.09345*, 2017.

[32] Shuang Liu, Olivier Bousquet, and Kamalika Chaudhuri. Approximation and convergence properties of generative adversarial learning. In *Advances in Neural Information Processing Systems*, pages 5551–5559, 2017.

[33] Zhihe Lu, Zhihang Li, Jie Cao, Ran He, and Zhenan Sun. Recent progress of face image synthesis. *arXiv preprint arXiv:1706.04717*, 2017.

[34] Rahul Mazumder, Trevor Hastie, and Robert Tibshirani. Spectral regularization algorithms for learning large incomplete matrices. *Journal of machine learning research*, 11(Aug):2287–2322, 2010.

[35] Patrick E McKnight, Katherine M McKnight, Souraya Sidani, and Aurelio Jose Figueredo. *Missing data: A gentle introduction*. Guilford Press, 2007.

[36] Mehdi Mirza and Simon Osindero. Conditional generative adversarial nets. *arXiv preprint arXiv:1411.1784*, 2014.

[37] Vaishnavh Nagarajan and J Zico Kolter. Gradient descent gan optimization is locally stable. In *Advances in Neural Information Processing Systems*, pages 5591–5600, 2017.

[38] Fulufhelo V Nelwamondo, Shakir Mohamed, and Tshilidzi Marwala. Missing data: A comparison of neural network and expectation maximization techniques. *Current Science*, pages 1514–1521, 2007.

[39] Sai Rajeswar, Sandeep Subramanian, Francis Dutil, Christopher Pal, and Aaron Courville. Adversarial generation of natural language. *arXiv preprint arXiv:1705.10929*, 2017.

[40] MATLAB Release. The mathworks. *Inc., Natick, Massachusetts, United States*, 488, 2013.

[41] Thomas Schlegl, Philipp Seeböck, Sebastian M Waldstein, Ursula Schmidt-Erfurth, and Georg Langs. Unsupervised anomaly detection with generative adversarial networks to guide marker discovery. In *International Conference on Information Processing in Medical Imaging*, pages 146–157. Springer, 2017.

[42] Ikaro Silva, George Moody, Daniel J Scott, Leo A Celi, and Roger G Mark. Predicting in-hospital mortality of icu patients: The physionet/computing in cardiology challenge 2012. In *Computing in Cardiology (CinC), 2012*, pages 245–248. IEEE, 2012.

[43] Luciana O Silva and Luis E Zárate. A brief review of the main approaches for treatment of missing data. *Intelligent Data Analysis*, 18(6):1177–1198, 2014.

[44] Nitish Srivastava, Geoffrey Hinton, Alex Krizhevsky, Ilya Sutskever, and Ruslan Salakhutdinov. Dropout: A simple way to prevent neural networks from overfitting. *The Journal of Machine Learning Research*, 15(1):1929–1958, 2014.

[45] Werner Wothke. Longitudinal and multigroup modeling with missing data. 2000.

[46] Jinsung Yoon, James Jordon, and Mihaela van der Schaar. Gain: Missing data imputation using generative adversarial nets. *arXiv preprint arXiv:1806.02920*, 2018.

[47] Lantao Yu, Weinan Zhang, Jun Wang, and Yong Yu. Seqgan: Sequence generative adversarial nets with policy gradient. In *AAAI*, pages 2852–2858, 2017.

[48] Kaiping Zheng, Jinyang Gao, Kee Yuan Ngiam, Beng Chin Ooi, and Wei Luen James Yip. Resolving the bias in electronic medical records. In *Proceedings of the 23rd ACM SIGKDD International Conference on Knowledge Discovery and Data Mining*, pages 2171–2180. ACM, 2017.

